# Random Walk Approach to Regret Minimization

**Hariharan Narayanan**
MIT
Cambridge, MA 02139
har@mit.edu

**Alexander Rakhlin**
University of Pennsylvania
Philadelphia, PA 19104
rakhlin@wharton.upenn.edu

## Abstract

We propose a computationally efficient random walk on a convex body which rapidly mixes to a time-varying Gibbs distribution. In the setting of online convex optimization and repeated games, the algorithm yields low regret and presents a novel efficient method for implementing mixture forecasting strategies.

## 1 Introduction

This paper brings together two topics: *online convex optimization* and *sampling from logconcave distributions over convex bodies*.

Online convex optimization has been a recent focus of research [30, 25], for it presents an abstraction that unifies and generalizes a number of existing results in online learning. Techniques from the theory of optimization (in particular, Fenchel and minimax duality) have proven to be key for understanding the rates of growth of regret [25, 1]. Deterministic *regularization* methods [3, 25] have emerged as natural black-box algorithms for regret minimization, and the choice of the regularization function turned out to play a pivotal role in limited-feedback problems [3]. In particular, the authors of [3] demonstrated the role of self-concordant regularization functions and the Dikin ellipsoid for minimizing regret. The latter gives a handle on the local geometry of the convex set, crucial for linear optimization with limited feedback.

Random walks in a convex body gained much attention following the breakthrough paper of Dyer, Frieze and Kannan [9], who exhibited a polynomial time randomized algorithm for estimating the volume of a convex body. It is known that the problem of computing this volume by a *deterministic* algorithm is #P-hard. Over the two decades following [9], the polynomial dependence of volume computation on the dimension $n$ has been drastically decreased from $O^*(n^{23})$ to $O^*(n^4)$ [17]. The development was accomplished through the study of several geometric random walks: the Ball Walk and Hit-and-Run (see [26] for a survey). The driving force behind such results are the isoperimetric inequalities which can be extended from uniform to general logconcave distributions. In particular, computing the volume of a convex body can be seen as a special case of integration of a logconcave function, and there has been a number of major results on mixing time for sampling from logconcave distributions [17, 18]. Connections to optimization have been established in [12, 18], among others. More recently, a novel random walk, called the *Dikin Walk* has been proposed in [19, 13]. By exploiting the local geometry of the set, this random walk is shown to mix rapidly, and offers a number of advantages over the other random walks.

While the aim of online convex optimization is different from that of sampling from logconcave distributions, the fact that the two communities recognized the importance of the Dikin ellipsoid is remarkable. In this paper we build a bridge between the two topics. We show that the problem of online convex optimization can be solved by sampling from logconcave distributions, and that the Dikin Walk can be adapted to mix rapidly to a certain time-varying distribution. In fact, it mixes fast enough that for linear cost functions *only one step* of the guided Dikin Walk is necessary per round of the repeated game. This is surprisingly similar to the sufficiency of one Damped Newton step of Algorithm 2 in [3], due to locally quadratic convergence ensured by the self-concordant regularizer.

The time-varying Gibbs distributions from which we sample are closely related to Mixture Forecasters and Bayesian Model Averaging methods (see [7, Section 11.10] as well as [29, 28, 4, 10]). To the best of our knowledge, the method presented in this paper is the first provably computationally-efficient approach to solving a class of problems which involves integrating over continuous sets of decisions. From the Bayesian point of view, our algorithm is an efficient procedure for sampling from posterior distributions, and can be used for settings outside of regret minimization.

**Prior work:**  The closest to our work is the result of [11] for Universal Portfolios. Unlike our one-step Markov chain, the algorithm of [11] works with a discretization of the probability simplex and requires a number of steps which has adverse dependence on the time horizon and accuracy. This seems unavoidable with the Grid Walk. In [2], it was shown that the Weighted Average Forecaster [15, 27] on a prohibitively large class of experts is optimal in terms of regret for a certain multitask problem, yet computationally inefficient. A Markov chain has been proposed with the required stationary distribution, but no mixing time bounds have been derived. In [8], the authors faced a similar problem whereby a near-optimal regret can be achieved by the Weighted Average Forecaster on a prohibitively large discretization of the set of decisions. Sampling from time-varying Markov chains has been investigated in the context of network dynamics [24], and has been examined from the point of view of linear stochastic approximation in reinforcement learning [14]. Beyond [11], we are not aware of any results to date where a provably rapidly mixing walk is used to solve regret minimization problems.

It is worth emphasizing that without the Dikin Walk [19], the one-step mixing results of this paper seem out of reach. In particular, when sampling from exponential distributions, the known bounds for the conductance of the Ball Walk and Hit-and-Run are not scale-independent. In order to obtain $O(\sqrt{T})$ regret, one has to be able to sample the target distribution with an error that is $O(1/\sqrt{T})$. As a consequence of the deterioration of the bounds on the conductance as the scale tends to zero, the number of steps necessary per round would tend to infinity as $T$ tends to infinity.

## 2   Main Results

Let $\mathcal{K} \subset \mathbb{R}^n$ be a convex compact set and let $\mathcal{F}$ be a set of convex functions from $\mathcal{K}$ to $\mathbb{R}$. *Online convex optimization* is defined as a repeated $T$-round game between the player (the algorithm) and Nature (adversary) [30, 25]. From the outset we assume that Nature is oblivious (see [7]), i.e. the individual sequence of decisions $\ell_1, \ldots, \ell_T \in \mathcal{F}$ can be fixed before the game. We are interested in randomized algorithms, and hence we consider the following online learning model: on round $t$, the player chooses a *distribution* (or, a *mixed strategy*) $\mu_{t-1}$ supported on $\mathcal{K}$ and "plays" a random $X_t \sim \mu_{t-1}$. Nature then reveals the cost function $\ell_t \in \mathcal{F}$. The goal of the player is to control *expected regret* (see Lemma 1) with respect to a randomized strategy defined by a fixed distribution $p_U \in \mathcal{P}$ for some collection of distributions $\mathcal{P}$. If $\mathcal{P}$ contains Dirac delta distributions, the comparator term is indeed the best fixed decision $x^* \in \mathcal{K}$ chosen in hindsight. A procedure which guarantees sublinear growth of regret for any distribution $p_U \in \mathcal{P}$ will be called *Hannan consistent* with respect to $\mathcal{P}$. We now state a natural procedure for updating distributions $\mu_t$ which guarantees Hannan consistency for a wide range of problems. This procedure is similar to the Mixture Forecaster used in the prediction context [29, 28, 4, 10]. Denote the cumulative cost functions by $L_t(x) = \sum_{s=1}^{t} \ell_s(x)$, with $L_0(x) \equiv 0$, and let $\eta > 0$ be a learning rate. Let $q_0(x)$ be some prior probability distribution supported on $\mathcal{K}$. Define the following sequence of functions

$$q_t(x) = q_0(x) \exp\left\{-\eta L_t(x)\right\}, \quad \forall t \in \{1, \ldots, T\} \tag{1}$$

for every $x \in \mathcal{K}$. Define the probability distribution $\mu_t$ over $\mathcal{K}$ at time $t$ to have density

$$\frac{d\mu_t(x)}{dx} = \frac{q_0(x)e^{-\eta L_t(x)}}{Z_t} \quad \text{where} \quad Z_t = \int_{x \in \mathcal{K}} q_t(x)dx. \tag{2}$$

Let $D(p||q)$ stand for the Kullback-Leibler divergence between distributions $p$ and $q$. The following lemma[1] gives an *equality* for expected regret with respect to a fixed randomized strategy. It bears

striking similarity to upper bounds on regret in terms of Bregman divergences for the Follow the Regularized Leader and Mirror Descent methods [23, 5], [7, Therem 11.1].

**Lemma 1.** *Let $X_t$ be a random variable distributed according to $\mu_{t-1}$, for all $t \in \{1, \ldots, T\}$, as defined in* (2). *Let $U$ be a random variable with distribution $p_U$. The expected regret is*

$$\mathbb{E}\left[\sum_{t=1}^{T} \ell_t(X_t) - \sum_{t=1}^{T} \ell_t(U)\right] = \eta^{-1}\left(D(p_U||\mu_0) - D(p_U||\mu_T)\right) + \eta^{-1}\sum_{t=1}^{T} D(\mu_{t-1}||\mu_t).$$

*Specializing to the case $\ell(x) \in [0, 1]$ over $\mathcal{K}$,*

$$\mathbb{E}\left[\sum_{t=1}^{T} \ell_t(X_t) - \sum_{t=1}^{T} \ell_t(U)\right] \leq \eta^{-1} D(p_U||\mu_0) + T\eta/8.$$

Before proceeding, let us make a few remarks. First, if the divergence between the comparator distribution $p_U$ and the prior $\mu_0$ is bounded, the result yields $O(\sqrt{T})$ rates of regret growth for bounded losses by choosing $\eta$ appropriately. To bound the divergence between a continuous initial $\mu_0$ and a point comparator at some $x^*$, the analysis can be carried out in two stages: comparison to a "small-covariance" Gaussian centered at $x^*$, followed by an observation that the loss of the "small-covariance" Gaussian strategy is not very different from the loss of the deterministic strategy $x^*$. This analysis can be found in [7, p. 326] and gives a near-optimal $O(\sqrt{T \log T})$ regret bound.

We also note that for linear cost functions, the notion of expected regret coincides with regret for deterministic strategies. Third, we note that if the prior is of the form $q_0(x) \propto \exp\{-R(x)\}$ for some convex function $R$, then $q_t(x) \propto \exp\{-(\eta L_t(x) + R(x))\}$, bearing similarity to the objective function of the Follow the Regularized Leader algorithm [23, 3]. In general, we can encode prior knowledge in $q_0$. For instance, if the cost functions are linear and the set $\mathcal{K}$ is a convex hull of $N$ vertices (e.g. probability simplex), then the minimum loss is attained at one of the vertices, and a uniform prior on the vertices yields the Weighted Average Forecaster with the usual $\log N$ dependence [7]. Finally, we note that in online convex optimization, one of the difficulties is the issue of projections back to the set $\mathcal{K}$. This issue does not arise when dealing with distributions, but instead translates into the *difficulty of sampling*. We find these parallels between sampling and optimization intriguing.

We defer the easy proof of Lemma 1 to p. 8. Having a bound on regret, a natural question is whether there exists a computationally efficient algorithm for playing $X_t$ according to the mixed strategy given in (2). The main result of this paper is that for linear Lipschitz cost functions the guided random walk (Algorithm 1 below) produces a sequence of points $X_1, \ldots, X_T \in \mathcal{K}$ with respective distributions $\sigma_0, \ldots, \sigma_{T-1}$ such that $\sigma_i$ is close to $\mu_i$ for all $0 \leq i \leq T-1$. Moreover, $X_i$ is obtained from $X_{i-1}$ with only one random step. The step requires sampling from a Gaussian distribution with covariance given by the Hessian of the self-concordant barrier and can be implemented efficiently whenever the Hessian can be computed. The computation time exactly matches [3, Algorithm 2]: it is the same as time spent inverting a Hessian matrix, which is $O(n^3)$ or less.

Let us now discuss our assumptions. First, the analysis of the random walk is carried out only for linear cost functions with a bounded Lipschitz constant. An analysis for general convex functions might be possible, but for the sake of brevity we restrict ourselves to the linear case. Note that convex cost functions can be linearized and a standard argument shows that regret for linearized functions can only be larger than that for the convex functions [30]. The second assumption is that $q_0$ does not depend on $T$ and has a bounded $L_2$ norm with respect to the uniform distribution on $\mathcal{K}$. This means that $q_0$ can be not only uniform, but, for instance, of the form $q_0(x) \propto \exp\{-R(x)\}$.

**Theorem 2.** *Suppose $\ell_t : \mathcal{K} \mapsto [0, 1]$ are linear functions with Lipschitz constant $1$ and the prior $q_0$ is of bounded $L_2$ norm with respect to uniform distribution on $\mathcal{K}$. Then the one-step random walk (Algorithm 1) produces a sequence $X_1, \ldots, X_T$ with distributions $\sigma_0, \ldots, \sigma_{T-1}$ such that for all $i$,*

$$\int_{x \in \mathcal{K}} |d\sigma_i(x) - d\mu_i(x)| \leq C\eta n^3 \nu^2,$$

*where $\mu_i$ are defined in* (2), *$\nu$ is the parameter of self-concordance, and $C$ is an absolute constant. Therefore, regret of Algorithm 1 is within $O(\sqrt{T})$ from the ideal procedure of Lemma 1. In*

*particular, by choosing $\eta$ appropriately, for an absolute constant $C'$,*

$$\mathbb{E}\left[\sum_{t=1}^{T}\ell_t(X_t) - \sum_{t=1}^{T}\ell_t(U)\right] \leq C'n^{3/2}\nu\sqrt{TD(p_U||\mu_0)}. \tag{3}$$

*Proof.* The statement follows directly from Lemma 1, Theorem 9, and an observation that for bounded losses

$$\left|\mathbb{E}_{\mu_{t-1}}\ell_t(X_t) - \mathbb{E}_{\sigma_{t-1}}\ell_t(X_t)\right| \leq \int_{x\in\mathcal{K}}|\ell_t(x)|\cdot|d\mu_{t-1}(x) - d\sigma_{t-1}(x)| \leq C\eta n^3\nu^2 \ .$$

$\square$

## 3   Sampling from a time-varying Gibbs distribution

**Sketch of the Analysis**   The sufficiency of only one step of the random walk is made possible by the fact that the distributions $\mu_{t-1}$ and $\mu_t$ are close, and thus $\mu_{t-1}$ is a (very) warm start for $\mu_t$. The reduction in distance between the distributions after a single step is due to a general fact (Lovász-Simonovits [16]) which we state in Theorem 6. The majority of the work goes into lower bounding the conductance of the random walk by a quantity *independent of $T$* (Lemma 5). Since the random walk of Algorithm 1 takes advantage of the local geometry of the set, the conductance is lower bounded by (a) proving an isoperimetric inequality (Theorem 3) for the Riemannian metric (which states that the measure of the gap between two well-separated sets is large) and (b) by proving that for close-by (in the Riemannian metric) points, their transition functions are not too different (Lemma 4). Section 3 is organized as follows. In Section 3.1, the main building blocks for proving mixing time are stated, and their proofs appear later in Section 4. In Section 3.2, we use the mixing result of Section 3.1 to show that Algorithm 1 indeed closely tracks the distributions $\mu_t$ (Theorem 9).

### 3.1   Bounding Mixing Time

In the remainder of this paper, $C$ will denote a universal constant that may change from line to line. For any function $F$ on the interior $int(\mathcal{K})$ having continuous derivatives of order $k$, for vectors $h_1,\ldots,h_k \in \mathbb{R}^n$ and $x \in int(\mathcal{K})$, for $k \geq 1$, we recursively define

$$D^k F(x)[h_1,\ldots,h_k] := \lim_{\epsilon\to 0}\frac{D^{k-1}(x+\epsilon h_k)[h_1,\ldots,h_{k-1}] - D^{k-1}(x)[h_1,\ldots,h_{k-1}]}{\epsilon},$$

where $D^0 F(x) := F(x)$. Let $F$ be a self-concordant barrier of $\mathcal{K}$ with a parameter $\nu$ (see [20]). For $x, y \in \mathcal{K}$, $\rho(x,y)$ is the distance in the Riemannian metric whose metric tensor is the Hessian of $F$. Thus, the metric tensor on the tangent space at $x$ assigns to a vector $v$ the length $\|v\|_x := D^2 F(x)[v,v]$, and to a pair of vectors $v, w$, the inner product $\langle v, w\rangle_x := D^2 F(x)[v,w]$. We have $\rho(x,y) = \inf_\Gamma \int_z \|d\Gamma\|_z$ where the infimum is taken over all rectifiable paths $\Gamma$ from $x$ to $y$. Let $\mathcal{M}$ be the metric space whose point set is $\mathcal{K}$ and metric is $\rho$. We assume $\ell_i$ are linear and $1-$Lipschitz with respect to $\rho$. For $x \in int(\mathcal{K})$, let $G_x$ denote the unique Gaussian probability density function on $\mathbb{R}^n$ such that

$$G_x(y) \propto \exp\left(-\frac{n\|x-y\|_x^2}{r^2} + V(x)\right), \quad \text{where} \quad V(x) = \frac{1}{2}\ln\det D^2 F(x) \ \text{and} \ r = 1/(Cn)$$

Further, define the scaled cumulative cost as $s_t(y) := r^2\eta L_t(y)$. Note that shape of $G_x$ is precisely given by the Dikin ellipsoid, which is defined as a unit ball in $\|\cdot\|_x$ around a point $x$ [20, 3].

The Markov chain $\mathcal{M}_t$ considered in this paper is such that for $x, y \in \mathcal{K}$, one step $x \to y$ is given by Algorithm 1. A simple calculation shows that the detailed balance conditions are satisfied with respect to a stationary distribution $\mu_t$ (defined in Eq. (2)). Therefore the Markov chain is reversible and has this stationary measure. The next results imply that this Markov chain is rapidly mixing. The first main ingredient is an isoperimetric inequality necessary for lower bounding conductance.

**Theorem 3.** *Let $S_1$ and $S_2$ be measurable subsets of $\mathcal{K}$ and $\mu$ a probability measure supported on $\mathcal{K}$ that possesses a density whose logarithm is concave. Then,*

$$\mu((\mathcal{K}\setminus S_1)\setminus S_2) \geq \frac{1}{2(1+3\nu)}\rho(S_1,S_2)\mu(S_1)\mu(S_2).$$

**Algorithm 1** One Step Random Walk $(X_t, s_t)$

---

**Input**: current point $X_t \in \mathcal{K}$ and scaled cumulative cost $s_t$.

**Output**: next point $X_{t+1} \in \mathcal{K}$

Toss a fair coin. **If** Heads, set $X_{t+1} := X_t$.

**Else**,

Sample $Z$ from $G_{X_t}$. If $Z \notin \mathcal{K}$, let $X_{t+1} := X_t$.

If $Z \in \mathcal{K}$, let

$$X_{t+1} := \begin{cases} Z & \text{with prob. } \min\left(1, \frac{G_Z(X_t)\exp(s_t(X_t))}{G_{X_t}(Z)\exp(s_t(Z))}\right) \\ X_t & \text{otherwise.} \end{cases}$$

---

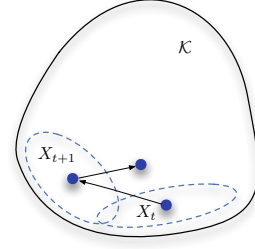

Figure 1: The new point is sampled from a Gaussian distribution whose shape is defined by the local metric. Dotted lines are the unit Dikin ellipsoids.

The next Lemma relates the Riemannian metric $\rho$ to the Markov Chain. Intuitively, it says that for close-by points, their transition distributions cannot be far apart.

**Lemma 4.** *If $x, y \in \mathcal{K}$ and $\rho(x, y) \le \frac{r}{C\sqrt{n}}$, then $d_{TV}(P_x, P_y) \le 1 - \frac{1}{C}$.*

Theorem 3 and Lemma 4 together give a lower bound on conductance of the Markov Chain.

**Lemma 5** (Bound on Conductance). *Let $\mu$ be any exponential distribution on $\mathcal{K}$. The conductance*

$$\Phi := \inf_{\mu(S_1) \le \frac{1}{2}} \frac{\int_{S_1} P_x(\mathcal{K} \setminus S_1) d\mu(x)}{\mu(S_1)}$$

*of the Markov Chain in Algorithm 1 is bounded below by $\frac{1}{C\nu n\sqrt{n}}$.*

The lower bound on conductance of Lemma 5 can now be used with the following general result on the reduction of distance between distributions.

**Theorem 6** (Lovász-Simonovits [16]). *Let $\gamma_0$ be the initial distribution for a lazy reversible ergodic Markov chain whose conductance is $\Phi$ and stationary measure is $\gamma$, and $\gamma_k$ be the distribution of the $k^{th}$ step. Let $M := \sup_S \frac{\gamma_0(S)}{\gamma(S)}$ where the supremum is over all measurable subsets $S$ of $\mathcal{K}$. For every bounded $f$, let $\|f\|_{2,\gamma}$ denote $\sqrt{\int_{\mathcal{K}} f(x)^2 d\gamma(x)}$. For any fixed $f$, let $Ef$ be the map that takes $x$ to $\int_{\mathcal{K}} f(y)dP_x(y)$. Then if $\int_{\mathcal{K}} f(x)d\gamma(x) = 0$,*

$$\|E^k f\|_{2,\gamma} \le \left(1 - \frac{\Phi^2}{2}\right)^k \|f\|_{2,\gamma}.$$

In summary, Lemma 5 provides a lower bound on conductance, while Theorem 6 ensures reduction of the norm whenever conductance is large enough. In the next section, these two are put together. We will show that reduction in the norm guarantees that the distribution after one step of the random walk ($k = 1$ in Theorem 6) is close to the desired distribution $\mu_t$.

## 3.2 Tracking the distributions

Let $\{\sigma_i\}_{i=1}^{\infty}$ be the probability measures with bounded density, supported on $\mathcal{K}$, corresponding to the distribution of a point during different steps of the evolution of the algorithm. For $i \in \mathbb{N}$, let $\|\cdot\|_{\mu_i}$ denote the $\mathcal{L}_2$ norm with respect to the measure $\mu_i$. We shall write $\|\cdot\|_i$ for brevity. Hence, for a measurable function $f : \mathcal{K} \to \mathbb{R}$, $\|f\|_i = \left(\int_{\mathcal{K}} f^2 d\mu_i\right)^{1/2}$. Furthermore,

$$\sup_{x \in \mathcal{K}} \frac{d\mu_i(x)}{d\mu_{i+1}(x)} = \sup_{x \in \mathcal{K}} \frac{q_0(x)e^{-\eta L_i(x)}dx}{q_0(x)e^{-\eta L_{i+1}(x)}dx} \frac{Z_{t+1}}{Z_t} \le e^{2\eta} \le 1 + \bar{\eta} \tag{4}$$

where we used the fact that $\ell_{i+1}(x) \le 1$ and $\bar{\eta}$ is an appropriate multiple of $\eta$, e.g. $\bar{\eta} = (e^2 - 1)\eta$ does the job. Analogously, $d\mu_{i+1}/d\mu_i \le 1 + \bar{\eta}$ over $\mathcal{K}$. It then follows that the norms at time $i$ and $i + 1$ are comparable:

$$\|f\|_i(1 + \bar{\eta})^{-1} \le \|f\|_{i+1} \le \|f\|_i(1 + \bar{\eta}) \tag{5}$$

The mixing results of Lemma 5 together with Theorem 6 imply

**Corollary 7.** *For any $i$,*

$$\left\|\frac{d\sigma_{i+1}}{d\mu_i} - 1\right\|_i \le \left\|\frac{d\sigma_i}{d\mu_i} - 1\right\|_i \left(1 - \left(\frac{1}{Cn^3\nu^2}\right)\right)$$

Corollary 7 says that $\sigma_{i+1}$ is "closer" than $\sigma_i$ to $\mu_i$ by a multiplicative constant. We now show that the distance of $\sigma_{i+1}$ to $\mu_{i+1}$ is (additively) not much worse than its distance to $\mu_i$. The multiplicative reduction in distance is shown to be dominating the additive increase, concluding the proof that $\sigma_i$ is close to $\mu_i$ for all $i$ (Theorem 9).

**Lemma 8.** *For any $i$, it holds that*

$$\left\|\frac{d\sigma_{i+1}}{d\mu_{i+1}} - 1\right\|_{i+1} \le (1+\bar\eta)^2 \left\|\frac{\sigma_{i+1}}{d\mu_i} - 1\right\|_i + \bar\eta(1+\bar\eta).$$

*Proof.*

$$\left\|\frac{d\sigma_{i+1}}{d\mu_{i+1}} - 1\right\|_{i+1} - \left\|\frac{d\sigma_{i+1}}{d\mu_i} - 1\right\|_i = \left\|\frac{d\sigma_{i+1}}{d\mu_{i+1}} - 1\right\|_{i+1} - \left\|\frac{d\sigma_{i+1}}{d\mu_i} - 1\right\|_{i+1} \qquad (6)$$

$$+ \left\|\frac{d\sigma_{i+1}}{d\mu_i} - 1\right\|_{i+1} - \left\|\frac{d\sigma_{i+1}}{d\mu_i} - 1\right\|_i. \qquad (7)$$

We first establish a bound of $C\eta$ on (6). For any function $f : \mathcal{K} \to \mathbb{R}$, let $f^+(x) = \max(0, f(x))$ and $f^-(x) = \min(0, f(x))$. By the triangle inequality,

$$\left\|\frac{d\sigma_{i+1}}{d\mu_{i+1}} - 1\right\|_{i+1} - \left\|\frac{d\sigma_{i+1}}{d\mu_i} - 1\right\|_{i+1} \le \left\|\frac{d\sigma_{i+1}}{d\mu_{i+1}} - \frac{d\sigma_{i+1}}{d\mu_i}\right\|_{i+1}.$$

Now, using (4) and (5),

$$\left\|\frac{d\sigma_{i+1}}{d\mu_{i+1}} - \frac{d\sigma_{i+1}}{d\mu_i}\right\|_{i+1}^2 = \left\|\left(\frac{d\sigma_{i+1}}{d\mu_{i+1}} - \frac{d\sigma_{i+1}}{d\mu_i}\right)^+\right\|_{i+1}^2 + \left\|\left(\frac{d\sigma_{i+1}}{d\mu_{i+1}} - \frac{d\sigma_{i+1}}{d\mu_i}\right)^-\right\|_{i+1}^2$$

$$\le \left\|\frac{d\sigma_{i+1}}{d\mu_i}\bar\eta\mathbf{1}\left[1 \ge \frac{d\mu_i}{d\mu_{i+1}}\right]\right\|_{i+1}^2 + \left\|\frac{d\sigma_{i+1}}{d\mu_i}\bar\eta\mathbf{1}\left[1 < \frac{d\mu_i}{d\mu_{i+1}}\right]\right\|_{i+1}^2$$

$$= \bar\eta^2\left\|\frac{d\sigma_{i+1}}{d\mu_i}\right\|_{i+1}^2 \le \bar\eta^2(1+\bar\eta)^2\left\|\frac{d\sigma_{i+1}}{d\mu_i}\right\|_i^2.$$

Thus, (6) is bounded as

$$\left\|\frac{d\sigma_{i+1}}{d\mu_{i+1}} - 1\right\|_{i+1} - \left\|\frac{d\sigma_{i+1}}{d\mu_i} - 1\right\|_{i+1} \le \bar\eta(1+\bar\eta)\left\|\frac{d\sigma_{i+1}}{d\mu_i}\right\|_i = \bar\eta(1+\bar\eta)\left(1 + \left\|\frac{\sigma_{i+1}}{d\mu_i} - 1\right\|_i\right)$$

Next, a bound on (7) follows simply by the norm comparison inequality (5):

$$\left\|\frac{d\sigma_{i+1}}{d\mu_i} - 1\right\|_{i+1} - \left\|\frac{d\sigma_{i+1}}{d\mu_i} - 1\right\|_i \le \bar\eta\left\|\frac{d\sigma_{i+1}}{d\mu_i} - 1\right\|_i.$$

The statement follows by rearranging the terms. □

**Theorem 9.** *If $\left\|\frac{d\sigma_0}{d\mu_0} - 1\right\|_0 < \bar\eta(1+\bar\eta)$, where $\bar\eta = (e^2 - 1)\eta$, then for all $i$,*

$$\left\|\frac{d\sigma_i}{d\mu_i} - 1\right\|_i \le C\eta n^3\nu^2.$$

*Consequently, for all $i$*

$$\int_{x\in\mathcal{K}} |d\sigma_i(x) - d\mu_i(x)| \le C\eta n^3\nu^2.$$

*Proof.* By Corollary 7 and Lemma 8, we see that

$$\left\|\frac{d\sigma_{i+1}}{d\mu_{i+1}} - 1\right\|_{i+1} \leq (1 + \bar{\eta})^2 \left(1 - \left(\frac{1}{Cn^3\nu^2}\right)\right) \left\|\frac{d\sigma_i}{d\mu_i} - 1\right\|_i + \bar{\eta}(1 + \bar{\eta}).$$

Since $\bar{\eta} = o(\frac{1}{n^3\nu^2})$,

$$\left\|\frac{d\sigma_{i+1}}{d\mu_{i+1}} - 1\right\|_{i+1} \leq \left(1 - \left(\frac{1}{Cn^3\nu^2}\right)\right) \left\|\frac{d\sigma_i}{d\mu_i} - 1\right\|_i + C\eta. \tag{8}$$

Let $0 \leq a < 1$ and $b > 0$, and $x_0, x_1, \ldots,$ be any sequence of non-negative numbers such that, $x_0 \leq b$ and for each $i$, $x_{i+1} \leq ax_i + b$. We see, by unfolding the recurrence, that $x_{i+1} \leq \frac{b}{1-a}$. From this and (8), the first statement of the theorem follows. The second statement follows from

$$\int |d\sigma_i - d\mu_i| = \int \left|\frac{d\sigma_i}{d\mu_i} - 1\right| d\mu_i \leq \left(\int \left(\frac{d\sigma_i}{d\mu_i} - 1\right)^2 d\mu_i\right)^{1/2} = \left\|\frac{d\sigma_i}{d\mu_i} - 1\right\|_i.$$

$\square$

# 4 Proof Sketch

In this section, we prove the main building blocks stated in Section 3.1. Consider a time step $t$. Let $d_{TV}$ represent total variation distance. Without loss of generality, assume $x$ is the origin and assume $s_t(x) = 0$. For $x \in \mathcal{K}$ and a vector $v$, $|v|_x$ is defined to be $\sup_{x \pm \alpha v \in \mathcal{K}} \alpha$. The following relation holds:

**Theorem 10** (Theorem 2.3.2 (iii) [21])**.** *Let F be a self-concordant barrier whose self-concordance parameter is $\nu$. Then $|h|_x \leq \|h\|_x \leq 2(1 + 3\nu)|h|_x$ for all $h \in \mathbb{R}^n$ and $x \in int(\mathcal{K})$.*

We term $(S_1, (\mathcal{M} \setminus S_1) \setminus S_2, S_2)$ a $\delta$-partition of $\mathcal{M}$, if $\delta \leq d_{\mathcal{M}}(S_1, S_2) := \inf_{x \in S_1, y \in S_2} d_{\mathcal{M}}(x, y)$, where $S_1$, $S_2$ are measurable subsets of $\mathcal{M}$. Let $\mathcal{P}_\delta$ be the set of all $\delta$-partitions of $\mathcal{M}$. If $\mu$ is a measure on $\mathcal{M}$, the isoperimetric constant is defined as

$$\mathcal{C}(\delta, \mathcal{M}, \mu) := \inf_{\mathcal{P}_\delta} \frac{\mu((\mathcal{M} \setminus S_1) \setminus S_2)}{\mu(S_1)\mu(S_2)} \quad \text{and} \quad \mathcal{C}_t := \mathcal{C}\left(\frac{r}{\sqrt{n}}, \mathcal{M}, \mu_t\right).$$

Given interior points $x, y$ in $int(\mathcal{K})$, suppose $p, q$ are the ends of the chord in $\mathcal{K}$ containing $x, y$ and $p, x, y, q$ lie in that order. Denote by $\sigma(x, y)$ the cross ratio $\frac{|x-y||p-q|}{|p-x||q-y|}$. Let $d_H$ denote the Hilbert (projective) metric defined by $d_H(x, y) := \ln(1 + \sigma(x, y))$. For two sets $S_1$ and $S_2$, let $\sigma(S_1, S_2) := \inf_{x \in S_1, y \in S_2} \sigma(x, y)$.

***Proof of Theorem 3****.* For any $z$ on the segment $\overline{xy}$ an easy computation shows that $d_H(x, z) + d_H(z, y) = d_H(x, y)$. Therefore it suffices to prove the result infinitesimally. By a result due to Nesterov and Todd [22, Lemma 3.1],

$$\|x - y\|_x - \|x - y\|_x^2 \leq \rho(x, y) \leq -\ln(1 - \|x - y\|_x). \tag{9}$$

whenever $\|x - y\|_x < 1$. From (9) $\lim_{y \to x} \frac{\rho(x, y)}{\|x - y\|_x} = 1$, and a direct computation shows that

$$\lim_{y \to x} \frac{d_H(x, y)}{|x - y|_x} = \lim_{y \to x} \frac{\sigma(x, y)}{|x - y|_x} \geq 1.$$

Hence, using Theorem 10, the Hilbert metric and the Riemannian metric satisfy

$$\rho(x, y) \leq 2(1 + 3\nu)d_H(x, y).$$

The statement of the theorem is now an immediate consequence of the following result due to Lovász and Vempala [18]: If $S_1$ and $S_2$ are measurable subsets of $\mathcal{K}$ and $\mu$ a probability measure supported on $\mathcal{K}$ that possesses a density whose logarithm is concave, then

$$\mu((\mathcal{K} \setminus S_1) \setminus S_2) \geq \sigma(S_1, S_2)\mu(S_1)\mu(S_2).$$

$\square$

**Proof of Lemma 5.** Let $S_1$ be a measurable subset of $\mathcal{K}$ such that $\mu(S_1) \leq \frac{1}{2}$ and $S_2 := \mathcal{K} \setminus S_1$ be its complement. Let $S_1' = S_1 \cap \{x | P_x(S_2) \leq 1/C\}$ and $S_2' = S_2 \cap \{y | P_y(S_1) \leq 1/C\}$. By the reversibility of the chain, which is easily checked,

$$\int_{S_1} P_x(S_2) d\mu(x) = \int_{S_2} P_y(S_1) d\mu(y).$$

If $x \in S_1'$ and $y \in S_2'$ then,

$$d_{TV}(P_x, P_y) := 1 - \int_{\mathcal{K}} \min\left(\frac{dP_x}{d\mu}(w), \frac{dP_y}{d\mu}(w)\right) d\mu(w) = 1 - \frac{1}{C}.$$

Lemma 4 implies that if $\rho(x, y) \leq \frac{r}{C\sqrt{n}}$, then $d_{TV}(P_x, P_y) \leq 1 - \frac{1}{C}$. Therefore

$$\rho(S_1', S_2') := \inf_{x \in S_1', y \in S_2'} \rho(x, y) \geq \frac{r}{C\sqrt{n}}. \tag{10}$$

Therefore Theorem 3 implies that

$$\mu((\mathcal{K} \setminus S_1') \setminus S_2') \geq \frac{\rho(S_1', S_2')}{2(1 + 3\nu)} \min(\mu(S_1'), \mu(S_2')) \geq \frac{r}{C\nu\sqrt{n}} \min(\mu(S_1'), \mu(S_2')).$$

First suppose $\mu(S_1') \geq (1 - \frac{1}{C})\mu(S_1)$ and $\mu(S_2') \geq (1 - \frac{1}{C})\mu(S_2)$. Then,

$$\int_{S_1} P_x(S_2) d\mu(x) \geq \mu((\mathcal{K} \setminus S_1') \setminus S_2') \geq \frac{\mathcal{C}\mu(S_1')}{C} \geq \frac{\mathcal{C}\min(\mu(S_1'), \mu(S_2'))}{C}$$

and we are done. Otherwise, without loss of generality, suppose $\mu(S_1') \leq (1 - \frac{1}{C})\mu(S_1)$. Then

$$\int_{S_1} P_x(S_2) d\mu(x) \geq \frac{\mu(S_1)}{C}$$

and we are done.

$\square$

**Proof of Lemma 1.** We have that

$$D(\mu_{t-1}||\mu_t) = \int_{\mathcal{K}} d\mu_{t-1} \log \frac{q_{t-1}Z_t}{Z_{t-1}q_t} = \log \frac{Z_t}{Z_{t-1}} + \int_{\mathcal{K}} \eta\ell_t(x) d\mu_{t-1}(x) = \log \frac{Z_t}{Z_{t-1}} + \eta\mathbb{E}\ell_t(X_t). \tag{11}$$

Rearranging, canceling the telescoping terms, and using the fact that $Z_0 = 1$

$$\eta\mathbb{E}\sum_{t=1}^{T} \ell_t(X_t) = \sum_{t=1}^{T} D(\mu_{t-1}||\mu_t) - \log Z_T.$$

Let $U$ be a random variable with a probability distribution $p_U$. Then

$$-\sum_{t=1}^{T} \mathbb{E}\ell_t(U) = \eta^{-1}\int_{\mathcal{K}} -\eta L_T(u) dp_U(u) = \eta^{-1}\int_{\mathcal{K}} dp_U(u) \log \frac{q_T(u)}{q_0(u)}$$

Combining,

$$\mathbb{E}\left[\sum_{t=1}^{T} \ell_t(X_t) - \sum_{t=1}^{T} \ell_t(U)\right] = \eta^{-1}\int_{\mathcal{K}} dp_U(u) \log \frac{q_T(u)/Z_T}{q_0(u)} + \eta^{-1}\sum_{t=1}^{T} D(\mu_{t-1}||\mu_t)$$

$$= \eta^{-1}(D(p_U||\mu_0) - D(p_U||\mu_T)) + \eta^{-1}\sum_{t=1}^{T} D(\mu_{t-1}||\mu_t).$$

Now, from Eq. (11), the KL divergence can be also written as

$$D(\mu_{t-1}||\mu_t) = \log \frac{\int_{\mathcal{K}} e^{-\eta\ell_t(x)} q_{t-1}(x) dx}{\int_{\mathcal{K}} q_{t-1}(x) dx} + \eta\mathbb{E}\ell_t(X_t) = \log \mathbb{E}e^{-\eta(\ell_t(X_t) - \mathbb{E}\ell_t(X_t))}$$

By representing the divergence in this form, one can obtain upper bounds via known methods, such as *Log-Sobolev inequalities* (e.g. [6]). In the simplest case of bounded loss, it is easy to show that $D(\mu_{t-1}||\mu_t) \leq O(\eta^2)$, and the particular constant $1/8$ can be obtained by, for instance, applying Lemma A.1 in [7]. This proves the second part of the lemma. $\square$

## Footnotes

[1]Due to its simplicity, the lemma has likely appeared in the literature, yet we could not locate a reference for this form with equality and in the context of online convex optimization. The closest results appear in [28, 10], [7, p. 326] in the context of prediction, and in [4] in the context of density estimation with exponential families.

# References

[1] J. Abernethy, A. Agarwal, P. L. Bartlett, and A. Rakhlin. A stochastic view of optimal regret through minimax duality. In *COLT '09*, 2009.

[2] J. Abernethy, P. L. Bartlett, and A. Rakhlin. Multitask learning with expert advice. In *Proceedings of The Twentieth Annual Conference on Learning Theory*, pages 484–498, 2007.

[3] J. Abernethy, E. Hazan, and A. Rakhlin. Competing in the dark: An efficient algorithm for bandit linear optimization. In *Proceedings of The Twenty First Annual Conference on Learning Theory*, 2008.

[4] K. S. Azoury and M. K. Warmuth. Relative loss bounds for on-line density estimation with the exponential family of distributions. *Machine Learning*, 43(3):211–246, June 2001.

[5] A. Beck and M. Teboulle. Mirror descent and nonlinear projected subgradient methods for convex optimization. *Oper. Res. Lett.*, 31(3):167–175, 2003.

[6] S. Boucheron, G. Lugosi, and P. Massart. Concentration inequalities using the entropy method. 31:1583–1614, 2003.

[7] N. Cesa-Bianchi and G. Lugosi. *Prediction, Learning, and Games*. Cambridge University Press, 2006.

[8] V. Dani, T. P. Hayes, and S. Kakade. The price of bandit information for online optimization. In *Advances in Neural Information Processing Systems 20*. Cambridge, MA, 2008.

[9] M. Dyer, A. Frieze, and R. Kannan. A random polynomial-time algorithm for approximating the volume of convex bodies. *Journal of the ACM (JACM)*, 38(1):1–17, 1991.

[10] S. Kakade and A. Ng. Online bounds for Bayesian algorithms. In *Proceedings of Neural Information Processing Systems (NIPS 17)*, 2005.

[11] A. Kalai and S. Vempala. Efficient algorithms for universal portfolios. *The Journal of Machine Learning Research*, 3:440, 2003.

[12] A.T. Kalai and S. Vempala. Simulated annealing for convex optimization. *Mathematics of Operations Research*, 31(2):253–266, 2006.

[13] R. Kannan and H. Narayanan. Random walks on polytopes and an affine interior point method for linear programming. In *STOC*, 2009. Accepted (pending revisions), Mathematics of Operations Research.

[14] V. R. Konda and J. N. Tsitsiklis. Linear stochastic approximation driven by slowly varying markov chains. *Systems and Control Letters*, 50:95–102, 2003.

[15] N. Littlestone and M. K. Warmuth. The weighted majority algorithm. *Information and Computation*, 108(2):212–261, 1994.

[16] L. Lovász and M. Simonovits. Random walks in a convex body and an improved volume algorithm. *Random Structures and Algorithms*, 4(4):359–412, 1993.

[17] L. Lovász and S. Vempala. Simulated annealing in convex bodies and an $o^*(n^4)$ volume algorithm. *J. Comput. Syst. Sci.*, 72(2):392–417, 2006.

[18] L. Lovász and S. Vempala. The geometry of logconcave functions and sampling algorithms. *Random Struct. Algorithms*, 30(3):307–358, 2007.

[19] H. Narayanan. Randomized interior point methods for sampling and optimization. *CoRR*, abs/0911.3950, 2009.

[20] A.S. Nemirovskii. Interior point polynomial time methods in convex programming, 2004.

[21] Y. E. Nesterov and A. S. Nemirovskii. *Interior Point Polynomial Algorithms in Convex Programming*. SIAM, Philadelphia, 1994.

[22] Y.E. Nesterov and MJ Todd. On the Riemannian geometry defined by self-concordant barriers and interior-point methods. *Foundations of Computational Mathematics*, 2(4):333–361, 2008.

[23] A. Rakhlin. Lecture notes on online learning, 2008. http://stat.wharton.upenn.edu/~rakhlin/papers/online_learning.pdf.

[24] D. Shah and J. Shin. Dynamics in congestion games. In *Proceedings of ACM Sigmetrics*, 2010.

[25] S. Shalev-Shwartz and Y. Singer. Convex repeated games and fenchel duality. In *NIPS*. 2007.

[26] S. Vempala. Geometric random walks: A survey. *In Combinatorial and computational geometry. Math. Sci. Res. Inst. Publ*, 52:577–616, 2005.

[27] V. Vovk. Aggregating strategies. In *Proceedings of the Third Annual Workshop on Computational Learning Theory*, pages 372–383. Morgan Kaufmann, 1990.

[28] V. Vovk. Competitive on-line statistics. *International Statistical Review*, 69:213–248, 2001.

[29] K. Yamanishi. Minimax relative loss analysis for sequential prediction algorithms using parametric hypotheses. In *COLT' 98*, pages 32–43, New York, NY, USA, 1998. ACM.

[30] M. Zinkevich. Online convex programming and generalized infinitesimal gradient ascent. In *ICML*, 2003.

